# Mapping Classifier Systems Into Neural Networks

Lawrence Davis

BBN Laboratories

BBN Systems and Technologies Corporation

10 Moulton Street

Cambridge, MA 02238

January 16, 1989

## Abstract

Classifier systems are machine learning systems incorporating a genetic algorithm as the learning mechanism. Although they respond to inputs that neural networks can respond to, their internal structure, representation formalisms, and learning mechanisms differ markedly from those employed by neural network researchers in the same sorts of domains. As a result, one might conclude that these two types of machine learning formalisms are intrinsically different. This is one of two papers that, taken together, prove instead that classifier systems and neural networks are equivalent. In this paper, half of the equivalence is demonstrated through the description of a transformation procedure that will map classifier systems into neural networks that are isomorphic in behavior. Several alterations on the commonly-used paradigms employed by neural network researchers are required in order to make the transformation work. These alterations are noted and their appropriateness is discussed. The paper concludes with a discussion of the practical import of these results, and with comments on their extensibility.

## 1  Introduction

Classifier systems are machine learning systems that have been developed since the 1970s by John Holland and, more recently, by other members of the genetic algorithm research community as well[1]. Classifier systems are varieties of genetic algorithms — algorithms for optimization and learning. Genetic algorithms employ techniques inspired by the process of biological evolution in order to "evolve" better and better

individuals that are taken to be solutions to problems such as optimizing a function, traversing a maze, etc. (For an explanation of genetic algorithms, the reader is referred to [Goldberg 1989].) Classifier systems receive messages from an external source as inputs and organize themselves using a genetic algorithm so that they will "learn" to produce responses for internal use and for interaction with the external source.

This paper is one of two papers exploring the question of the formal relationship between classifier systems and neural networks. As normally employed, the two sorts of algorithms are probably distinct, although a procedure for translating the operation of neural networks into isomorphic classifier systems is given in [Belew and Gherrity 1988]. The technique Belew and Gherrity use does not include the conversion of the neural network learning procedure into the classifier system framework, and it appears that the technique will not support such a conversion. Thus, one might conjecture that the two sorts of machine learning systems employ learning techniques that cannot be reconciled, although if there were a subsumption relationship, Belew and Gherrity's result suggests that the set of classifier systems might be a superset of the set of neural networks.

The reverse conclusion is suggested by consideration of the inputs that each sort of learning algorithm processes. When viewed as "black boxes", both mechanisms for learning receive inputs, carry out self-modifying procedures, and produce outputs. The class of inputs that are traditionally processed by classifier systems — the class of bit strings of a fixed length — is a subset of the class of inputs that have been traditionally processed by neural networks. Thus, it appears that classifier systems operate on a subset of the inputs that neural networks can process, when viewed as mechanisms that can modify their behavior.

In fact, both these impressions are correct. One can translate classifier systems into neural networks, preserving their learning behavior, and one can translate neural networks into classifier systems, again preserving learning behavior. In order to do so, however, some specializations of each sort of algorithm must be made. This paper deals with the translation from classifier systems to neural networks and with those specializations of neural networks that are required in order for the translation to take place. The reverse translation uses quite different techniques, and is treated in [Davis 1989].

The following sections contain a description of classifier systems, a description of the transformation operator, discussions of the extensibility of the proof, comments on some issues raised in the course of the proof, and conclusions.

## 2   Classifier Systems

A classifier system operates in the context of an *environment* that sends messages to the system and provides it with reinforcement based on the behavior it displays. A classifier system has two components — a *message list* and a population of rule-like entities called *classifiers*. Each message on the message list is composed of bits, and

each has a pointer to its source (messages may be generated by the environment or by a classifier.) Each classifier in the population of classifiers has three components: a *match string* made up of the characters 0,1, and # (for "don't care"); a *message* made up of the characters 0 and 1; and a *strength*. The top-level description of a classifier system is that it contains a population of production rules that attempt to match some condition on the message list (thus "classifying" some input) and post their message to the message list, thus potentially affecting the environment or other classifiers. Reinforcement from the environment is used by the classifier system to modify the strengths of its classifiers. Periodically, a genetic algorithm is invoked to create new classifiers, which replace certain members of the classifier set. (For an explanation of classifier systems, their potential as machine learning systems, and their formal properties, the reader is referred to [Holland et al 1986].)

Let us specify these processing stages more precisely. A classifier system operates by cycling through a fixed list of procedures. In order, these procedures are:

**Message List Processing**. 1. Clear the message list. 2. Post the environmental messages to the message list. 3. Post messages to the message list from classifiers in the post set of the previous cycle. 4. Implement environmental reinforcement by analyzing the messages on the message list and altering the strength of classifiers in the post set of the previous cycle.

**Form the Bid Set**. 1. Determine which classifiers match a message in the message list. A classifier matches a message if each bit in its match field matches its corresponding message bit. A 0 matches a 0, a 1 matches a 1, and a # matches either bit. The set of all matching classifiers forms the current *bid set*. 2. Implement bid taxes by subtracting a portion of the strength of each classifier $c$ in the bid set. Add the strength taken from $c$ to the strength of the classifier or classifiers that posted messages matched by $c$ in the prior step.

**Form the Post Set**. 1. If the bid set is larger than the maximum post set size, choose classifiers stochastically to post from the bid set, weighting them in proportion to the magnitude of their bid taxes. The set of classifiers chosen is the *post set*.

**Reproduction** Reproduction generally does not occur on every cycle. When it does occur, these steps are carried out: 1. Create $n$ children from parents. Use crossover and/or mutation, choosing parents stochastically but favoring the strongest ones. (Crossover and mutation are two of the operators used in genetic algorithms.) 2. Set the strength of each child to equal the average of the strength of that child's parents. (Note: this is one of many ways to set the strength of a new classifier. The transformation will work in analogous ways for each of them.) 3. Remove $n$ members of the classifier population and add the $n$ new children to the classifier population.

# 3   Mapping Classifiers Into Classifier Networks

The mapping operator that I shall describe maps each classifier into a *classifier network*. Each classifier network has links to environmental input units, links to

other classifier networks, and match, post, and message units. The weights on the links leading to a match node and leaving a post node are related to the fields in the match and message lists in the classifier. An additional link is added to provide a bias term for the match node. (Note: it is assumed here that the environment posts at most one message per cycle. Modifications to the transformation operator to accommodate multiple environmental messages are described in the final comments of this paper.)

Given a classifier system CS with $n$ classifiers, each matching and sending messages of length $m$, we can construct an isomorphic neural network composed of $n$ classifier networks in the following way. For each classifier $c$ in CS, we construct its corresponding classifier network, composed of $n$ match nodes, 1 post node, and $m$ message nodes. One match node (the *environmental match node*) has links to inputs from the environment. Each of the other match nodes is linked to the message and post node of another classifier network. The reader is referred to Figure 2 for an example of such a transformation.

Each match node in a classifier network has $m + 1$ incoming links. The weights on the first m links are derived by applying the following transformation to the m elements of c's match field: 0 is associated with weight $-1$, 1 is associated with weight 1, and # is associated with weight 0. The weight of the final link is set to $m + 1 - l$, where $l$ is the number of links with weight $= 1$. Thus, a classifier with match field (1 0 # 0 1) would have an associated network with weights on the links leading to its match nodes of 1, -1, 0, -1, 1, and 4. A classifier with match field (1 0 #) would have weights of 1, -1, 0, and 3.

The weights on the links to each message node in the classifier network are set to equal the corresponding element of the classifier's message field. Thus, if the message field of the classifier were (0 1 0), the weights on the links leading to the three message nodes in the corresponding classifier network would be 0, 1, and 0. The weights on all other links in the classifier network are set to 1.

Each node in a classifier network uses a threshold function to determine its activation level. Match nodes have thresholds $= m + .9$. All other nodes have thresholds $= .9$. If a node's threshold is exceeded, the node's activation level is set to 1. If not, it is set to 0.

Each classifier network has an associated quantity called strength that may be altered when the network is run, during the processing cycle described below.

A cycle of processing of a classifier system CS maps onto the following cycle of processing in a set of classifier networks:

**Message List Processing.** 1. Compute the activation level of each message node in CS. 2. If the environment supplies reinforcement on this cycle, divide that reinforcement by the number of post nodes that are currently active, plus 1 if the environment posted a message on the preceding cycle, and add the quotient to the strength of each active post node's classifier network. 3. If there is a message on this cycle from the environment, map it onto the first $m$ environment nodes so that each node associated with a 0 is off and each node associated with a 1 is on. Turn the final environmental node on. If there is no environmental message, turn all environmental

nodes off.

**Form the Bid Set.** 1. Compute the activation level of each match node in each classifier network. 2. Compute the activation level of each bid node in each classifier network (the set of classifier networks with an active bid node is the bid set). 3. Subtract a fixed proportion of the strength of each classifier network cn in the bid set. Add this amount to the strength of those networks connected to an active match node in cn. (Strength given to the environment passes out of the system.)

**Form the Post Set.** 1. If the bid set is larger than the maximum post set size, choose networks stochastically to post from the bid set, weighting them in proportion to the magnitude of their bid taxes. The set of networks chosen is the *post set*. (This might be viewed as a stochastic n-winners-take-all procedure).

**Reproduction.** If this is a cycle on which reproduction would occur in the classifier system, carry out its analog in the neural network in the following way. 1. Create $n$ children from parents. Use crossover and/or mutation, choosing parents stochastically but favoring the strongest ones. The ternary alphabet composed of $-1$, 1, and 0 is used instead of the classifier alphabet of 0, 1, and #. After each operator is applied, the final member of the match list is set to $m + 1 - l$. 2. Write over the weights on the match links and the message links of $n$ classifier networks to match the weights in the children. Choose networks to be re-weighted stochastically, so that the weakest ones are most likely to be chosen. Set the strength of each re-weighted classifier network to be the average of the strengths of its parents.

It is simple to show that a classifier network match node will match a message in just those cases in which its associated classifier matched a message. There are three cases to consider. If the original match character was a #, then it matched any message bit. The corresponding link weight is set to 0, so the state of the node it comes from will not affect the activation of the match node it goes to. If the original match character was a 1, then its message bit had to be a 1 for the message to be matched. The corresponding link weight is set to 1, and we see by inspection of the weight on the final link, the match node threshold, and the fact that no other type of link has a positive weight, that every link with weight 1 must be connected to an active node for the match node to be activated. Finally, the link weight corresponding to a 0 is set to -1. If any of these links is connected to a node that is active, then the effect is that of turning off a node connected to a link with weight 1, and we have just seen that this will cause the match node to be inactive.

Given this correspondence in matching behavior, one can verify that a set of classifier networks associated with a classifier system has the following properties: During each cycle of processing of the classifier system, a classifier is in the bid set in just those cases in which its associated network has an active bid node. Assuming that both systems use the same randomizing technique, initialized in the same way, the classifier is in the post set in just those cases when the network is in the post set. Finally, the parents that are chosen for reproduction are the transformas of those chosen in the classifier system, and the children produced are the transformations of the classifier system parents. The two systems are isomorphic in operation, assuming that they use the same random number generator.

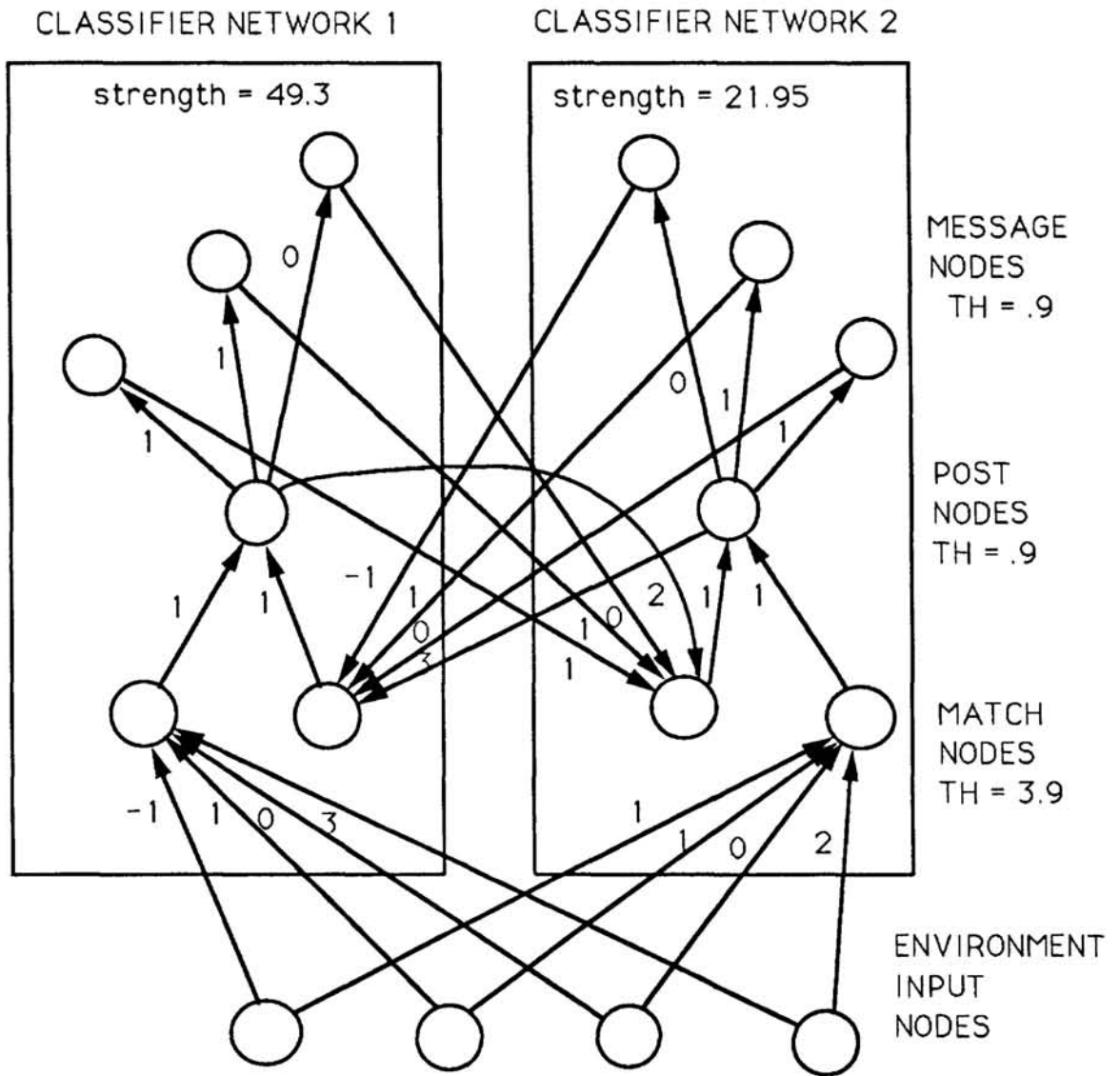

Figure 1: Result of mapping a classifier system
with two classifiers into a neural network.
Classifier 1 has match field (0 1 #), message field (1 1 0),
and strength 49.3. Classifier 2 has match field (1 1 #),
message field (0 1 1), and strength 21.95.

# 4  Concluding Comments

The transformation procedure described above will map a classifier system into a neural network that operates in the same way. There are several points raised by the techniques used to accomplish the mapping. In closing, let us consider four of them.

First, there is some excess complexity in the classifier networks as they are shown here. In fact, one could eliminate all non-environmental match nodes and their links, since one can determine whenever a classifier network is reweighted whether it matches the message of each other classifier network in the system. If so, one could introduce a link directly from the post node of the other classifier network to the post node of the new network. The match nodes to the environment are necessary, as long as one cannot predict what messages the environment will post. Message nodes are necessary as long as messages must be sent out to the environment. If not, they and their incoming links could be eliminated as well. These simplifications have not been introduced here because the extensions discussed next require the complexity of the current architecture.

Second, on the genetic algorithm side, the classifier system considered here is an extremely simple one. There are many extensions and refinements that have been used by classifier system researchers. I believe that such refinements can be handled by expanded mapping procedures and by modifications of the architecture of the classifier networks. To give an indication of the way such modifications would go, let us consider two sample cases. The first is the case of an environment that may produce multiple messages on each cycle. To handle multiple messages, an additional link must be added to each environmental match node with weight set to the match node's threshold. This link will *latch* the match node. An additional match node with links to the environment nodes must be added, and a latched counting node must be attached to it. Given these two architectural modifications, the cycle is modified as follows: During the message matching cycle, a series of subcycles is carried out, one for each message posted by the environment. In each subcycle, an environmental message is input and each environmental match node computes its activation. The environmental match nodes are latched, so that each will be active if it matched any environmental message. The count nodes will record how many were matched by each classifier network. When bid strength is paid from a classifier network to the posters of messages that it matched, the divisor is the number of environmental messages matched as recorded by the count node, plus the number of other messages matched. Finally, when new weights are written onto a classifier network's links, they are written onto the match node connected to the count node as well. A second sort of complication is that of *pass-through* bits — bits that are passed from a message that is matched to the message that is posted. This sort of mechanism can be implemented in an obvious fashion by complicating the structure of the classifier network. Similar complications are produced by considering multiple-message matching, negation, messages to effectors, and so forth. It is an open question whether all such cases can be handled by modifying the architecture and the mapping operator, but I have not yet found one that cannot be so handled.

Third, the classifier networks do not use the sigmoid activation functions that support hill-climbing techniques such as back-propagation. Further, they are recurrent networks rather than strict feed-forward networks. Thus, one might wonder whether the fact that one can carry out such transformations should affect the behavior of researchers in the field. This point is one that is taken up at greater length in the companion paper. My conclusion there is that several of the techniques imported into the neural network domain by the mapping appear to improve the performance of neural networks. These include tracking strength in order to guide the learning process, using genetic operators to modify the network makeup, and using population-level measurements in order to determine what aspects of a network to use in reproduction. The reader is referred to [Montana and Davis 1989] for an example of the benefits to be gained by employing these techniques.

Finally, one might wonder what the import of this proof is intended to be. In my view, this proof and the companion proof suggest some exciting ways in which one can hybridize the learning techniques of each field. One such approach and its successful application to a real-world problem is characterized in [Montana and Davis 1989].

## Footnotes

[1]This paper has benefited from discussions with Wayne Mesard, Rich Sutton, Ron Williams, Stewart Wilson, Craig Shaefer, David Montana, Gil Syswerda and other members of BARGAIN, the Boston Area Research Group in Genetic Algorithms and Inductive Networks.

# References

[1] Belew, Richard K. and Michael Gherrity, "Back Propagation for the Classifier System", in preparation.

[2] Davis, Lawrence, "Mapping Neural Networks into Classifier Systems", submitted to the 1989 International Conference on Genetic Algorithms.

[3] Goldberg, David E. *Genetic Algorithms in Search, Optimization, and Machine Learning*, Addison Wesley 1989.

[4] Holland, John H, Keith J. Holyoak, Richard E. Nisbett, and Paul R. Thagard, *Induction*, MIT Press, 1986.

[5] Montana, David J. and Lawrence Davis, "Training Feedforward Neural Networks Using Genetic Algorithms", submitted to the 1989 International Joint Conference on Artificial Intelligence.
